# A Method for the Efficient Design of Boltzmann Machines for Classification Problems

**Ajay Gupta and Wolfgang Maass***
Department of Mathematics, Statistics, and Computer Science
University of Illinois at Chicago
Chicago IL, 60680

## Abstract

We introduce a method for the efficient design of a Boltzmann machine (or a Hopfield net) that computes an arbitrary given Boolean function $f$. This method is based on an efficient simulation of acyclic circuits with threshold gates by Boltzmann machines. As a consequence we can show that various concrete Boolean functions $f$ that are relevant for classification problems can be computed by scalable Boltzmann machines that are guaranteed to converge to their global maximum configuration with high probability after constantly many steps.

## 1 INTRODUCTION

A Boltzmann machine ([AHS], [HS], [AK]) is a neural network model in which the units update their states according to a stochastic decision rule. It consists of a set $\mathcal{U}$ of units, a set $C$ of *unordered* pairs of elements of $\mathcal{U}$, and an assignment of connection strengths $S : C \rightarrow \mathbf{R}$. A configuration of a Boltzmann machine is a map $k : \mathcal{U} \rightarrow \{0,1\}$. The consensus $C(k)$ of a configuration $k$ is given by $C(k) = \sum_{\{u,v\} \in C} S(\{u,v\}) \cdot k(u) \cdot k(v)$. If the Boltzmann machine is currently in configuration $k$ and unit $u$ is considered for a state change, then the acceptance

*This paper was written during a visit of the second author at the Department of Computer Science of the University of Chicago.

probability for this state change is given by $\frac{1}{1+e^{-\Delta C/c}}$. Here $\Delta C$ is the change in the value of the consensus function $C$ that would result from this state change of $u$, and $c > 0$ is a fixed parameter (the "temperature").

Assume that $n$ units of a Boltzmann machine $B$ have been declared as input units and $m$ other units as output units. One says that $B$ computes a function $f : \{0,1\}^n \to \{0,1\}^m$ if for any clamping of the input units of $B$ according to some $\underline{a} \in \{0,1\}^n$ the only global maxima of the consensus function of the clamped Boltzmann machine are those configurations where the output units are in the states given by $f(\underline{a})$.

Note that even if one leaves the determination of the connection strengths for a Boltzmann machine up to a learning procedure ([AHS], [HS], [AK]), one has to know in advance the required number of hidden units, and how they should be connected (see section 10.4.3 of [AK] for a discussion of this open problem).

Ad hoc constructions of efficient Boltzmann machines tend to be rather difficult (and hard to verify) because of the cyclic nature of their "computations".

We introduce in this paper a new method for the construction of efficient Boltzmann machines for the computation of a given Boolean function $f$ (the same method can also be used for the construction of Hopfield nets). We propose to construct first an acyclic Boolean circuit $T$ with threshold gates that computes $f$ (this turns out to be substantially easier). We show in section 2 that any Boolean threshold circuit $T$ can be simulated by a Boltzmann machine $B(T)$ of the same size as $T$. Furthermore we show in section 3 that a minor variation of $B(T)$ is likely to converge very fast. In Section 4 we discuss applications of our method for various concrete Boolean functions.

## 2   SIMULATION OF THRESHOLD CIRCUITS BY BOLTZMANN MACHINES

A threshold circuit $T$ (see [M], [PS], [R], [HMPST]) is a labeled *acyclic directed graph*. We refer to the number of edges that are directed into (out of) a node of $T$ as the indegree (outdegree) of that node. Its nodes of indegree 0 are labeled by input variables $x_i (i \in \{1, \ldots, n\})$. Each node $g$ of indegree $l > 0$ in $T$ is labeled by some arbitrary Boolean threshold function $F_g : \{0,1\}^l \to \{0,1\}$, where $F_g(y_1, \ldots, y_l) = 1$ if and only if $\sum_{i=1}^{l} \alpha_i y_i \geq t$ (for some arbitrary parameters $\alpha_1, \ldots, \alpha_l, t \in \mathbf{R}$; w.l.o.g. $\alpha_1, \ldots, \alpha_l, t \in \mathbf{Z}$ [M]). One views such node $g$ as a threshold gate that computes $F_g$. If $m$ nodes of a threshold circuit $T$ are in addition labeled as output nodes, one defines in the usual manner the Boolean function $f : \{0,1\}^n \to \{0,1\}^m$ that is computed by $T$.

We simulate $T$ by the following Boltzmann machine $B(T) = <\mathcal{U}, \mathcal{C}, S>$ (note that $T$ has *directed* edges, while $B(T)$ has *undirected* edges). We reserve for each node $g$ of $T$ a separate unit $b(g)$ of $B(T)$. We set

$$\mathcal{U} := \{b(g) | g \text{ is a node of } T\} \text{ and}$$
$$\mathcal{C} := \{\{b(g'), b(g)\} | g', g \text{ are nodes of } T \text{ so that either } g' = g \text{ or } g', g \text{ are connected by an edge in } T\}.$$

Consider an arbitrary unit $b(g)$ of $B(T)$. We define the connection strengths $S(\{b(g)\})$ and $S(\{b(g'), b(g)\})$ (for edges $< g', g >$ of $T$) by induction on the length of the longest path in $T$ from $g$ to a node of $T$ with outdegree 0.

If $g$ is a gate of $T$ with outdegree 0 then we define $S(\{b(g)\}) := -2t + 1$, where t is the threshold of $g$, and we set $S(\{b(g'), b(g)\}) := 2\alpha(< g', g >)$ (where $\alpha(< g', g >)$ is the weight of the directed edge $< g', g >$ in $T$).

Assume that $g$ is a threshold gate of $T$ with outdegree $> 0$. Let $g_1, \ldots, g_k$ be the immediate successors of $g$ in $T$. Set $w := \sum_{i=1}^{k} |S(\{b(g), b(g_i)\})|$ (we assume that the connection strengths $S(\{b(g), b(g_i)\})$ have already been defined). We define $S(\{b(g)\}) := -(2w + 2) \cdot t + w + 1$, where $t$ is the threshold of gate $g$. Furthermore for every edge $< g', g >$ in $T$ we set $S(\{b(g'), b(g)\}) := (2w + 2) \cdot \alpha(< g', g >)$.

**Remark:** It is obvious that for problems in $TC^o$ (see section 4) the size of connection strengths in $B(T)$ can be bounded by a polynomial in $n$.

**Theorem 2.1** *For any threshold circuit $T$ the Boltzmann machine $B(T)$ computes the same Boolean function as $T$.*

**Proof of Theorem 2.1:**

Let $\underline{a} \in \{0, 1\}^n$ be an arbitrary input for circuit $T$. We write $g(\underline{a}) \in \{0, 1\}$ for the output of gate $g$ of $T$ for circuit input $\underline{a}$.

Consider the Boltzmann machine $B(T)_{\underline{a}}$ with the $n$ units $b(g)$ for input nodes $g$ of $T$ clamped according to $\underline{a}$. We show that the configuration $K_{\underline{a}}$ of $B(T)_{\underline{a}}$ where $b(g)$ is on if and only if $g(\underline{a}) = 1$ is the only global maximum (in fact: the only *local* maximum) of the consensus function $C$ for $B(T)_{\underline{a}}$.

Assume for a contradiction that configuration $K$ of $B(T)_{\underline{a}}$ is a global maximum of the consensus function $C$ and $K \neq K_{\underline{a}}$. Fix a node $g$ of $T$ of minimal depth in $T$ so that $K(b(g)) \neq K_{\underline{a}}(b(g)) = g(\underline{a})$. By definition of $B(T)_{\underline{a}}$ this node $g$ is not an input node of $T$. Let $K'$ result form $K$ by changing the state of $b(g)$. We will show that $C(K') > C(K)$, which is a contradiction to the choice of $K$.

We have (by the definition of $C$)

$$C(K') - C(K) = (1 - 2K(b(g))) \cdot (S_1 + S_2 + S(\{b(g)\})), \text{ where}$$
$$S_1 := \sum \{K(b(g')) \cdot S(\{b(g'), b(g)\})| < g', g > \text{ is an edge in } T\}$$
$$S_2 := \sum \{K(b(g')) \cdot S(\{b(g), b(g')\})| < g, g' > \text{ is an edge in } T\}.$$

Let $w$ be the parameter that occurs in the definition of $S(\{b(g)\})$ (set $w := 0$ if $g$ has outdegree 0). Then $|S_2| \leq w$. Let $p_1, \ldots, p_m$ be the immediate predecessors of $g$ in $T$, and let $t$ be the threshold of gate $g$. Assume first that $g(\underline{a}) = 1$. Then $S_1 = (2w + 2) \cdot \sum_{i=1}^{m} \alpha(< p_i, g >) \cdot p_i(\underline{a}) \geq (2w + 2) \cdot t$. This implies that $S_1 + S_2 > (2w + 2) \cdot t - w - 1$, and therefore $S_1 + S_2 + S(\{b(g)\}) > 0$, hence $C(K') - C(K) > 0$.

If $g(\underline{a}) = 0$ then we have $\sum_{i=1}^{m} \alpha(< p_i, g >) \cdot p_i(\underline{a}) \leq t - 1$, thus $S_1 = (2w + 2) \cdot \sum_{i=1}^{m} \alpha(< p_i, g >) \cdot p_i(\underline{a}) \leq (2w + 2) \cdot t - 2w - 2$. This implies that $S_1 + S_2 < (2w + 2) \cdot t - w - 1$, and therefore $S_1 + S_2 + S(\{b(g)\}) < 0$. We have in this case $K(b(g)) = 1$, hence $C(K') - C(K) = (-1) \cdot (S_1 + S_2 + S(\{b(g)\})) > 0$. $\square$

# 3   THE CONVERGENCE SPEED OF THE CONSTRUCTED BOLTZMANN MACHINES

We show that the constructed Boltzmann machines will converge relatively fast to a global maximum configuration. This positive result holds both if we view $B(T)$ as a sequential Boltzmann machine (in which units are considered for a state change one at a time), and if we view $B(T)$ as a parallel Boltzmann machine (where several units are simultaneously considered for a state change). In fact, it even holds for unlimited parallelism, where *every* unit is considered for a state change at every step. Although unlimited parallelism appears to be of particular interest in the context of brain models and for the design of massively parallel machines, there are hardly any positive results known for this case (see section 8.3 in [AK]).

If $g$ is a gate in $T$ with outdegree $> 1$ then the current state of unit $b(g)$ of $B(T)$ becomes relevant at several different time points (whenever one of the immediate successors of $g$ is considered for a state change). This effect increases the probability that unit $b(g)$ may cause an "error." Therefore the error probability of an output unit of $B(T)$ does not just depend on the number of nodes in $T$, but on the number $N(T)$ of nodes in a tree $T'$ that results if we replace in the usual fashion the directed graph of $T$ by a tree $T'$ of the same depth (one calls a directed graph a *tree* if all of its nodes have outdegree $\leq 1$).

To be precise, we define by induction on the depth of $g$ for each gate $g$ of $T$ a tree $\mathrm{Tree}(g)$ that replaces the subcircuit of $T$ below $g$. If $g_1, \ldots, g_k$ are the immediate predecessors of $g$ in $T$ then $\mathrm{Tree}(g)$ is the tree which has $g$ as root and $\mathrm{Tree}(g^1), \ldots, \mathrm{Tree}(g_k)$ as immediate subtrees (it is understood that if some $g_i$ has another immediate successor $g' \neq g$ then different copies of $\mathrm{Tree}(g_i)$ are employed in the definition of $\mathrm{Tree}(g)$ and $\mathrm{Tree}(g')$).

We write $|\mathrm{Tree}(g)|$ for the number of nodes in $\mathrm{Tree}(g)$ , and $N(T)$ for $\sum\{|\mathrm{Tree}(g)| \mid g$ is an output node of $T\}$. It is easy to see that if $T$ is synchronous (i.e. depth $(g'') = \mathrm{depth}(g') + 1$ for all edges $< g', g'' >$ in $T$) then $|\mathrm{Tree}(g)| \leq s^{d-1}$ for any node $g$ in $T$ of depth $d$ which has $s$ nodes in the subcircuit of $T$ below $g$. Therefore $N(T)$ is polynomial in $n$ if $T$ is of constant depth and polynomial size (this can be achieved for all problems in $TC^0$, see Section 4).

We write $B^\delta(T)$ for the variation of the Boltzmann machine $B(T)$ of section 2 where each connection strength in $B(T)$ is multiplied by $\delta$ ($\delta > 0$). Equivalently one could view $B^\delta(T)$ as a machine with the *same* connection strengths as $B(T)$ but a lower "temperature" (replace $c$ by $c/\delta$).

**Theorem 3.1** *Assume that $T$ is a threshold circuit of depth $d$ that computes a Boolean function $f : \{0,1\}^n \to \{0,1\}^m$. Let $B^\delta(T)_{\underline{a}}$ be the Boltzmann machine that results from clamping the input units of $B^\delta(T)$ according to $\underline{a}$ ($\underline{a} \in \{0,1\}^n$).*

*Assume that $0 = q_0 < q_1 < \ldots < q_d$ are arbitrary numbers such that for every $i \in \{1, \ldots, d\}$ and every gate $g$ of depth $i$ in $T$ the corresponding unit $b(g)$ is considered for a state change at some step during interval $(q_{i-1}, q_i]$. There is no restriction on how many other units are considered for a state change at any step.*

*Let $t$ be an arbitrary time step with $t \geq q_d$. Then the output units of $B(T)$ are at*

*the end of step t with probability $\geq 1 - N(T) \cdot \frac{1}{1+e^{\delta/c}}$ in the states given by $f(\underline{a})$.*

**Remarks:**

1. For $\delta := n$ this probability converges to 1 for $n \to \infty$ if $T$ is of constant depth and polynomial size.

2. The condition on the timing of state changes in Theorem 3.1 has been formulated in a very general fashion in order to make it applicable to all of the common types of Boltzmann machines.For a sequential Boltzmann machine (see [AK], section 8.2) one can choose $q_i - q_{i-1}$ sufficiently large (for example polynomially in the size of $T$) so that with high probability every unit of $B(T)$ is considered for a state change during the interval $(q_{i-1}, q_i]$. On the other hand, for a synchronous Boltzmann machine with limited parallelism ([AK], section 8.3) one may apply the result to the case where every unit $b(g)$ with $g$ of depth $i$ in $T$ is considered for a state change at step $i$ (set $q_i := i$). Theorem 3.1 also remains valid for unlimited parallelism ([AK], section 8.3), where every unit is considered for a state change at every step (set $q_i := i$). In fact, not even synchronicity is required for Theorem 3.1, and it also applies to asynchronous parallel Boltzmann machines ([AK], section 8.3.2).

3. For sequential Boltzmann machines in general the available upper bounds for their convergence speed are very unsatisfactory. In particular no upper bounds are known which are polynomial in the number of units (see section 3.5 of [AK]). For Boltzmann machines with unlimited parallelism one can in general not even prove that they converge to a global maximum of their consensus function (section 8.3 of [AK]).

**Proof of Theorem 3.1:** We prove by induction on $i$ that for every gate $g$ of depth $i$ in $T$ and every step $t \geq q_i$ the unit $b(g)$ is at the end of step $t$ with probability $\geq 1 - |\text{Tree}(g)| \cdot \frac{1}{1+e^{\delta/c}}$ in state $g(\underline{a})$.

Assume that $g_1, \ldots, g_k$ are the immediate predecessors of gate $g$ in $T$. By definition we have $|\text{Tree}(g)| = 1 + \sum_{j=1}^{k} |\text{Tree}(g_j)|$. Let $t' \leq t$ be the last step before $t$ at which $b(g)$ has been considered for a state change. Since $T \geq q_i$ we have $t' > q_{i-1}$. Thus for each $j = 1, \ldots, k$ we can apply the induction hypothesis to unit $b(g_j)$ and step $t' - 1 \geq q_{\text{depth}(g_j)}$. Hence with probability $\geq 1 - (|\text{Tree}(g)| - 1) \cdot \frac{1}{1+e^{\delta/c}}$ the state of the units $b(g_1), \ldots, b(g_k)$ at the end of step $t' - 1$ are $g_1(\underline{a}), \ldots, g_k(\underline{a})$. Assume now that the unit $b(g_j)$ is at the end of step $t' - 1$ in state $g_j(\underline{a})$, for $j = 1, \ldots, k$. If $g$ is at the beginning of step $t'$ not in state $g(\underline{a})$, then a state change of unit $b(g)$ would increase the consensus function by $\Delta C \geq \delta$ (independently of the current status of units $b(\tilde{g})$ for immediate successors $\tilde{g}$ of $g$ in $T$). Thus $b(g)$ accepts in this case the change to state $g(\underline{a})$ with probability $\frac{1}{1+e^{-\Delta C/c}} \geq \frac{1}{1+e^{-\delta/c}} = 1 - \frac{1}{1+e^{\delta/c}}$. On the other hand, if $b(g)$ is already at the beginning of step $t'$ in state $g(\underline{a})$, then a change of its state would decrease the consensus by at least $\delta$. Thus $b(g)$ remains with probability $\geq 1 - \frac{1}{1+e^{\delta/c}}$ in state $g(\underline{a})$. The preceding considerations imply that unit $b(g)$ is at the end of step $t'$ (and hence at the end of step $t$) with probability $\geq 1 - |\text{Tree}(g)| \cdot \frac{1}{1+e^{\delta/c}}$ in state $g(\underline{a})$. $\square$

# 4  APPLICATIONS

The complexity class $TC^0$ is defined as the class of all Boolean functions $f : \{0,1\}^* \to \{0,1\}^*$ for which there exists a family $(T_n)_{n \in N}$ of threshold circuits of some constant depth so that for each $n$ the circuit $T_n$ computes $f$ for inputs of length $n$, and so that the number of gates in $T_n$ and the absolute value of he weights of threshold gates in $T_n$ (all weights are assumed to be integers) are bounded by a polynomial in $n$ ([HMPST], [PS]).

**Corollary 4.1 (to Theorems 2.1, 3.1):** *Every Boolean function $f$ that belongs to the complexity class $TC^0$ can be computed by scalable (i.e. polynomial size) Boltzmann machines whose connection strengths are integers of polynomial size and which converge for state changes with unlimited parallelism with high probability in constantly many steps to a global maximum of their consensus function.*

The following Boolean functions are known to belong to the complexity class $TC^0$: AND, OR, PARITY; SORTING, ADDITION, SUBTRACTION, MULTIPLICATION and DIVISION of binary numbers; DISCRETE FOURIER TRANSFORM, and approximations to arbitrary analytic functions with a convergent rational power series ([CVS], [R], [HMPST]).

**Remarks:**

1. One can also use the method from this paper for the efficient construction of a Boltzmann machine $B_{\mathbf{p_1},\dots,\mathbf{p_k}}$ that can decide very fast to which of $k$ stored "patterns" $\mathbf{p_1},\dots,\mathbf{p_k} \in \{0,1\}^n$ the current input $\underline{x} \in \{0,1\}^n$ to the Boltzmann machine has the closest "similarity."

   For arbitrary fixed "patterns" $\mathbf{p_1}, \cdots, \mathbf{p_k} \in \{0,1\}^n$ let $f_{\mathbf{p_1},\dots,\mathbf{p_k}} : \{0,1\}^n \to \{0,1\}^k$ be the *pattern classification function* whose $i$th output bit is 1 if and only if the Hamming distance between the input $\underline{x} \in \{0,1\}^n$ and $\mathbf{p_i}$ is less or equal to the Hamming distance between $\underline{x}$ and $\mathbf{p_j}$, for all $j \neq i$.

   We write $HD(\underline{x},\underline{y})$ for the Hamming distance $\sum_{i=1}^{n} |x_i - y_i|$ of strings $\underline{x},\underline{y}, \in \{0,1\}^n$. One has $HD(\underline{x},\underline{y}) = \sum_{y_i=0} x_i + \sum_{y_i=1}(1 - x_i)$, and therefore $HD(\underline{x}, \mathbf{p_j}) - HD(\underline{x}, \mathbf{p_l}) = \sum_{i=1}^{n} \alpha_i x_i + c$ for suitable coefficients $\alpha_i \in \{-2,-1,0,1,2\}$ and $c \in \mathbf{Z}$ (that depend on the fixed patterns $\mathbf{p_j}, \mathbf{p_l} \in \{0,1\}^n$). Thus there is a threshold circuit that consists of a single threshold gate which outputs 1 if $HD(\underline{x}, \mathbf{p_j}) \leq HD(\underline{x}, \mathbf{p_l})$, and 0 otherwise.

   The function $f_{\mathbf{p_1},\dots,\mathbf{p_k}}$ can be computed by a threshold circuit $T$ of depth 2 whose $j$th output gate is the AND of $k-1$ gates as above which check for $l \in \{1,\dots,k\} - \{j\}$ whether $HD(\underline{x}, \mathbf{p_j}) \leq HD(\underline{x}, \mathbf{p_l})$ (note that the underlying graph of $T$ is the same for any choice of the patterns $\mathbf{p_1},\dots,\mathbf{p_k}$). The desired Boltzmann machine $B_{\mathbf{p_1},\dots,\mathbf{p_k}}$ is the Boltzmann machine $B(T)$ for this threshold circuit $T$.

2. Our results are also of interest in the context of *learning algorithms* for Boltzmann machines. For example, the previous remark provides a *single* graph $< \mathcal{U}, \mathcal{C} >$ of a Boltzmann machine with $n$ input units, $k$ output units, and $k^2 - k$ hidden units, that is able to compute with a suitable assignment of

connection strengths (that may arise from a learning algorithm for Boltzmann machines) *any* function $f_{\mathbf{p}_1,\ldots,\mathbf{p}_k}$ (for any choice of $\mathbf{p}_1,\ldots,\mathbf{p}_k \in \{0,1\}^n$).

Similarly we get from Theorem 2.1 together with a result from [M] the graph $< \mathcal{U},\mathcal{C} >$ of a Boltzmann machine with $n$ input units, $n$ hidden units, and one output unit, that can compute with a suitable assignment of connection strengths *any* symmetric function $f : \{0,1\}^n \to \{0,1\}$ ($f$ is called symmetric if $f(x_i,\cdots,x_n)$ depends only on $\sum_{i=1}^n x_i$; examples of symmetric functions are AND, OR, PARITY).

*Acknowledgment:* We would like to thank Georg Schnitger for his suggestion to investigate the convergence speed of the constructed Boltzmann machines.

# References

[AK]    E. Aarts, J. Korst, Simulated Annealing and Boltzmann Machines, John Wiley & Sons (New York, 1989).

[AHS]   D.H. Ackley, G.E. Hinton, T.J. Sejnowski, A learning algorithm for Boltzmann machines, Cognitive Science, 9, 1985, pp. 147-169.

[HS]    G.E. Hinton, T.J. Sejinowski, Learning and relearning in Boltzmann machines, in: D.E. Rumelhart, J.L McCelland, & the PDP Research Group (Eds.), Parallel Distributed Processing: Explorations in the Microstructure of Cognition, MIT Press (Cambridge, 1986), pp. 282-317.

[CVS]   A.K. Chandra, L.J. Stockmeyer, U. Vishkin, Constant depth reducibility, SIAM, J. Comp., 13, 1984, pp. 423-439.

[HMPST] A. Hajnal, W. Maass, P. Pudlak, M. Szegedy, G. Turan, Threshold circuits of bounded depth, to appear in J. of Comp. and Syst. Sci. (for an extended abstract see Proc. of the 28th IEEE Conf. on Foundations of Computer Science, 1987, pp.99-110).

[M]     S. Muroga, Threshold Logic and its Applications, John Wiley & Sons (New York, 1971).

[PS]    I. Parberry, G. Schnitger, Relating Boltzmann machines to conventional models of computation, Neural Networks, 2, 1989, pp. 59-67.

[R]     J. Reif, On threshold circuits and polynomial computation, Proc. of the 2nd Annual Conference on Structure in Complexity Theory, IEEE Computer Society Press, Washington, 1987, pp. 118-123.
